# Parallelized Stochastic Gradient Descent

**Martin A. Zinkevich**
Yahoo! Labs
Sunnyvale, CA 94089
`maz@yahoo-inc.com`

**Markus Weimer**
Yahoo! Labs
Sunnyvale, CA 94089
`weimer@yahoo-inc.com`

**Alex Smola**
Yahoo! Labs
Sunnyvale, CA 94089
`smola@yahoo-inc.com`

**Lihong Li**
Yahoo! Labs
Sunnyvale, CA 94089
`lihong@yahoo-inc.com`

## Abstract

With the increase in available data parallel machine learning has become an increasingly pressing problem. In this paper we present the first parallel stochastic gradient descent algorithm including a detailed analysis and experimental evidence. Unlike prior work on parallel optimization algorithms [5, 7] our variant comes with parallel acceleration guarantees and it poses no overly tight latency constraints, which might only be available in the multicore setting. Our analysis introduces a novel proof technique — contractive mappings to quantify the speed of convergence of parameter distributions to their asymptotic limits. As a side effect this answers the question of how quickly stochastic gradient descent algorithms reach the asymptotically normal regime [1, 8].

## 1   Introduction

Over the past decade the amount of available data has increased steadily. By now some industrial scale datasets are approaching Petabytes. Given that the bandwidth of storage and network per computer has not been able to keep up with the increase in data, the need to design data analysis algorithms which are able to perform most steps in a distributed fashion without tight constraints on communication has become ever more pressing. A simple example illustrates the dilemma. At current disk bandwidth and capacity (2TB at 100MB/s throughput) it takes at least 6 hours to read the content of a single harddisk. For a decade, the move from batch to online learning algorithms was able to deal with increasing data set sizes, since it reduced the runtime behavior of inference algorithms from cubic or quadratic to linear in the sample size. However, whenever we have more than a single disk of data, it becomes computationally infeasible to process all data by stochastic gradient descent which is an inherently sequential algorithm, at least if we want the result within a matter of hours rather than days.

Three recent papers attempted to break this parallelization barrier, each of them with mixed success. [5] show that parallelization is easily possible for the *multicore* setting where we have a tight coupling of the processing units, thus ensuring extremely low latency between the processors. In particular, for non-adversarial settings it is possible to obtain algorithms which scale perfectly in the number of processors, both in the case of bounded gradients and in the strongly convex case. Unfortunately, these algorithms are not applicable to a MapReduce setting since the latter is fraught with considerable latency and bandwidth constraints between the computers.

A more MapReduce friendly set of algorithms was proposed by [3, 9]. In a nutshell, they rely on distributed computation of gradients locally on each computer which holds parts of the data and subsequent aggregation of gradients to perform a global update step. This algorithm scales linearly

in the amount of data and log-linearly in the number of computers. That said, the overall cost in terms of computation and network is very high: it requires many passes through the dataset for convergence. Moreover, it requires many synchronization sweeps (i.e. MapReduce iterations). In other words, this algorithm is computationally very wasteful when compared to online algorithms.

[7] attempted to deal with this issue by a rather ingenious strategy: solve the sub-problems exactly on each processor and in the end average these solutions to obtain a joint solution. The key advantage of this strategy is that only a single MapReduce pass is required, thus dramatically reducing the amount of communication. Unfortunately their proposed algorithm has a number of drawbacks: the theoretical guarantees they are able to obtain imply a significant *variance* reduction relative to the single processor solution [7, Theorem 3, equation 13] but *no bias reduction whatsoever* [7, Theorem 2, equation 9] relative to a single processor approach. Furthermore, their approach requires a relatively expensive algorithm (a full batch solver) to run on each processor. A further drawback of the analysis in [7] is that the convergence guarantees are very much dependent on the degree of strong convexity as endowed by regularization. However, since regularization tends to decrease with increasing sample size the guarantees become increasingly loose in practice as we see more data.

We attempt to combine the benefits of a single-average strategy as proposed by [7] with asymptotic analysis [8] of online learning. Our proposed algorithm is strikingly simple: denote by $c^i(w)$ a loss function indexed by $i$ and with parameter $w$. Then each processor carries out stochastic gradient descent on the set of $c^i(w)$ with a fixed learning rate $\eta$ for $T$ steps as described in Algorithm 1.

---
**Algorithm 1** $\mathrm{SGD}(\{c^1, \ldots, c^m\}, T, \eta, w_0)$

---
   **for** $t = 1$ **to** $T$ **do**
      Draw $j \in \{1 \ldots m\}$ uniformly at random.
      $w_t \leftarrow w_{t-1} - \eta \partial_w c^j(w_{t-1})$.
   **end for**
   **return** $w_T$.

---

On top of the SGD routine which is carried out on each computer we have a master-routine which aggregates the solution in the same fashion as [7].

---
**Algorithm 2** $\mathrm{ParallelSGD}(\{c^1, \ldots c^m\}, T, \eta, w_0, k)$

---
   **for all** $i \in \{1, \ldots k\}$ **parallel do**
      $v_i = \mathrm{SGD}(\{c^1, \ldots c^m\}, T, \eta, w_0)$ on client
   **end for**
   Aggregate from all computers $v = \frac{1}{k} \sum_{i=1}^{k} v_i$ and **return** $v$

---

The key *algorithmic* difference to [7] is that the batch solver of the inner loop is replaced by a stochastic gradient descent algorithm which digests *not* a fixed fraction of data but rather a random fixed subset of data. This means that if we process $T$ instances per machine, each processor ends up seeing $\frac{T}{m}$ of the data which is likely to exceed $\frac{1}{k}$.

| Algorithm | Latency tolerance | MapReduce | Network IO | Scalability |
|---|---|---|---|---|
| Distributed subgradient [3, 9] | moderate | **yes** | high | **linear** |
| Distributed convex solver [7] | **high** | **yes** | **low** | unclear |
| Multicore stochastic gradient [5] | low | no | n.a. | **linear** |
| This paper | **high** | **yes** | **low** | **linear** |

A direct implementation of the algorithms above would place every example on every machine: however, if $T$ is much less than $m$, then it is only necessary for a machine to have access to the data it actually touches. Large scale learning, as defined in [2], is when an algorithm is bounded by the time available instead of by the amount of data available. Practically speaking, that means that one can consider the actual data in the real dataset to be a subset of a virtually infinite set, and drawing with replacement (as the theory here implies) and drawing without replacement on the

---

**Algorithm 3** SimuParallelSGD(Examples $\{c^1, \ldots c^m\}$, Learning Rate $\eta$, Machines $k$)

---

Define $T = \lfloor m/k \rfloor$
Randomly partition the examples, giving $T$ examples to each machine.
**for all** $i \in \{1, \ldots k\}$ **parallel do**
   Randomly shuffle the data on machine $i$.
   Initialize $w_{i,0} = 0$.
   **for all** $t \in \{1, \ldots T\}$: **do**
     Get the $t$th example on the $i$th machine (this machine), $c^{i,t}$
     $w_{i,t} \leftarrow w_{i,t-1} - \eta \partial_w c^i(w_{i,t-1})$
   **end for**
**end for**
Aggregate from all computers $v = \frac{1}{k} \sum_{i=1}^{k} w_{i,t}$ and **return** $v$.

---

infinite data set can both be simulated by shuffling the real data and accessing it sequentially. The initial distribution and shuffling can be a part of how the data is saved. SimuParallelSGD fits very well with the large scale learning paradigm as well as the MapReduce framework. Our paper applies an anytime algorithm via stochastic gradient descent. The algorithm requires no communication between machines until the end. This is perfectly suited to MapReduce settings. Asymptotically, the error approaches zero. The amount of time required is independent of the number of examples, only depending upon the regularization parameter and the desired error at the end.

## 2 Formalism

In stark contrast to the simplicity of Algorithm 2, its convergence analysis is highly technical. Hence we limit ourselves to presenting the main results in this extended abstract. Detailed proofs are given in the appendix. Before delving into details we briefly outline the proof strategy:

- When performing stochastic gradient descent with fixed (and sufficiently small) learning rate $\eta$ the distribution of the parameter vector is asymptotically normal [1, 8]. Since all computers are drawing from the same data distribution they all converge to the same limit.
- Averaging between the parameter vectors of $k$ computers reduces variance by $O(k^{-\frac{1}{2}})$ similar to the result of [7]. However, it does *not* reduce bias (this is where [7] falls short).
- To show that the bias due to joint initialization decreases we need to show that the *distribution* of parameters per machine converges sufficiently quickly to the limit distribution.
- Finally, we also need to show that the mean of the limit distribution for fixed learning rate is sufficiently close to the risk minimizer. That is, we need to take finite-size learning rate effects into account relative to the asymptotically normal regime.

### 2.1 Loss and Contractions

In this paper we consider estimation with convex loss functions $c^i : \ell_2 \to [0, \infty)$. While our analysis extends to other Hilbert Spaces such as RKHSs we limit ourselves to this class of functions for convenience. For instance, in the case of regularized risk minimization we have

$$c^i(w) = \frac{\lambda}{2}\|w\|^2 + L(x^i, y^i, w \cdot x^i) \tag{1}$$

where $L$ is a convex function in $w \cdot x^i$, such as $\frac{1}{2}(y^i - w \cdot x^i)^2$ for regression or $\log[1 + \exp(-y^i w \cdot x^i)]$ for binary classification. The goal is to find an approximate minimizer of the overall risk

$$c(w) = \frac{1}{m}\sum_{i=1}^{m} c^i(w). \tag{2}$$

To deal with *stochastic* gradient descent we need tools for quantifying distributions over $w$.

**Lipschitz continuity:** A function $f : \mathcal{X} \to \mathbf{R}$ is Lipschitz continuous with constant $L$ with respect to a distance $d$ if $|f(x) - f(y)| \leq Ld(x, y)$ for all $x, y \in \mathcal{X}$.

**Hölder continuity:** A function $f$ is Hölder continuous with constant $L$ and exponent $\alpha$ if $|f(x) - f(y)| \leq Ld^\alpha(x, y)$ for all $x, y \in \mathcal{X}$.

**Lipschitz seminorm:** [10] introduce a seminorm. With minor modification we use

$$\|f\|_{\text{Lip}} := \inf\{l \text{ where } |f(x) - f(y)| \leq ld(x, y) \text{ for all } x, y \in \mathcal{X}\}. \tag{3}$$

That is, $\|f\|_{\text{Lip}}$ is the smallest constant for which Lipschitz continuity holds.

**Hölder seminorm:** Extending the Lipschitz norm for $\alpha \geq 1$:

$$\|f\|_{\text{Lip}_\alpha} := \inf\{l \text{ where } |f(x) - f(y)| \leq ld^\alpha(x, y) \text{ for all } x, y \in \mathcal{X}\}. \tag{4}$$

**Contraction:** For a metric space $(M, d)$, $f : M \to M$ is a contraction mapping if $\|f\|_{\text{Lip}} < 1$.

In the following we assume that $\|L(x, y, y')\|_{\text{Lip}} \leq G$ as a function of $y'$ for all occurring data $(x, y) \in \mathcal{X} \times \mathcal{Y}$ and for all values of $w$ within a suitably chosen (often compact) domain.

**Theorem 1 (Banach's Fixed Point Theorem)** *If $(M, d)$ is a non-empty complete metric space, then any contraction mapping $f$ on $(M, d)$ has a unique fixed point $x^* = f(x^*)$.*

**Corollary 2** *The sequence $x_t = f(x_{t-1})$ converges linearly with $d(x^*, x_t) \leq \|f\|_{\text{Lip}}^t d(x_0, x^*)$.*

Our strategy is to show that the stochastic gradient descent mapping

$$w \leftarrow \phi^i(w) := w - \eta \nabla c^i(w) \tag{5}$$

is a contraction, where $i$ is selected uniformly at random from $\{1, \ldots m\}$. This would allow us to demonstrate exponentially fast convergence. Note that since the algorithm selects $i$ at random, different runs with the same initial settings can produce different results. A key tool is the following:

**Lemma 3** *Let $c^* \geq \|\partial_{\hat{y}} L(x^i, y^i, \hat{y})\|_{\text{Lip}}$ be a Lipschitz bound on the loss gradient. Then if $\eta \leq (\|x^i\|^2 c^* + \lambda)^{-1}$ the update rule (5) is a contraction mapping in $\ell_2$ with Lipschitz constant $1 - \eta\lambda$.*

We prove this in Appendix B. If we choose $\eta$ "low enough", gradient descent uniformly becomes a contraction. We define

$$\eta^* := \min_i \left(\|x^i\|^2 c^* + \lambda\right)^{-1}. \tag{6}$$

## 2.2 Contraction for Distributions

For fixed learning rate $\eta$ stochastic gradient descent is a Markov process with state vector $w$. While there is considerable research regarding the asymptotic properties of this process [1, 8], not much is known regarding the number of iterations required until the asymptotic regime is assumed. We now address the latter by extending the notion of contractions from mappings of points to mappings of distributions. For this we introduce the Monge-Kantorovich-Wasserstein earth mover's distance.

**Definition 4 (Wasserstein metric)** *For a Radon space $(M, d)$ let $P(M, d)$ be the set of all distributions over the space. The Wasserstein distance between two distributions $X, Y \in P(M, d)$ is*

$$W_z(X, Y) = \left[\inf_{\gamma \in \Gamma(X,Y)} \int_{x,y} d^z(x, y)\, d\gamma(x, y)\right]^{\frac{1}{z}} \tag{7}$$

*where $\Gamma(X, Y)$ is the set of probability distributions on $(M, d) \times (M, d)$ with marginals $X$ and $Y$.*

This metric has two very important properties: it is complete and a contraction in $(M, d)$ induces a contraction in $(P(M, d), W_z)$. Given a mapping $\phi : M \to M$, we can construct $\mathbf{p} : P(M, d) \to P(M, d)$ by applying $\phi$ pointwise to $M$. Let $X \in P(M, d)$ and let $X' := \mathbf{p}(X)$. Denote for any measurable event $E$ its pre-image by $\phi^{-1}(E)$. Then we have that $X'(E) = X(\phi^{-1}(E))$.

**Lemma 5** *Given a metric space $(M, d)$ and a contraction mapping $\phi$ on $(M, d)$ with constant $c$, $\mathbf{p}$ is a contraction mapping on $(P(M, d), W_z)$ with constant $c$.*

This is proven in Appendix C. This shows that any single mapping is a contraction. However, since we draw $c^i$ at random we need to show that a mixture of such mappings is a contraction, too. Here the fact that we operate on distributions comes handy since the mixture of mappings on distribution is a mapping on distributions.

**Lemma 6** *Given a Radon space $(M, d)$, if $\mathbf{p}_1 \ldots \mathbf{p}_k$ are contraction mappings with constants $c_1 \ldots c_k$ with respect to $W_z$, and $\sum_i a_i = 1$ where $a_i \geq 0$, then $\mathbf{p} = \sum_{i=1}^{k} a_i \mathbf{p}_i$ is a contraction mapping with a constant of no more than $[\sum_i a_i (c_i)^z]^{\frac{1}{z}}$.*

**Corollary 7** *If for all $i$, $c_i \leq c$, then $\mathbf{p}$ is a contraction mapping with a constant of no more than $c$.*

This is proven in Appendix C. We apply this to SGD as follows: Define $\mathbf{p}^* = \frac{1}{m} \sum_{i=1}^{m} \mathbf{p}^i$ to be the stochastic operation in one step. Denote by $D_\eta^0$ the initial parameter distribution from which $w_0$ is drawn and by $D_\eta^t$ the parameter distribution after $t$ steps, which is obtained via $D_\eta^t = \mathbf{p}^*(D_\eta^{t-1})$. Then the following holds:

**Theorem 8** *For any $z \in \mathbf{N}$, if $\eta \leq \eta^*$, then $\mathbf{p}^*$ is a contraction mapping on $(M, W_z)$ with contraction rate $(1 - \eta\lambda)$. Moreover, there exists a unique fixed point $D_\eta^*$ such that $\mathbf{p}^*(D_\eta^*) = D_\eta^*$. Finally, if $w_0 = 0$ with probability 1, then $W_z(D_\eta^0, D_\eta^*) = \frac{G}{\lambda}$, and $W_z(D_\eta^T, D_\eta^*) \leq \frac{G}{\lambda}(1 - \eta\lambda)^T$.*

This is proven in Appendix F. The contraction rate $(1 - \eta\lambda)$ can be proven by applying Lemma 3, Lemma 5, and Corollary 6. As we show later, $w_t \leq G/\lambda$ with probability 1, so $\Pr_{w \in D_\eta^*}[d(0, w) \leq G/\lambda] = 1$, and since $w_0 = 0$, this implies $W_z(D_\eta^0, D_\eta^*) = G/\lambda$. From this, Corollary 2 establishes $W_z(D_\eta^T, D_\eta^*) \leq \frac{G}{\lambda}(1 - \eta\lambda)^T$.

This means that for a suitable choice of $\eta$ we achieve exponentially fast convergence in $T$ to some stationary distribution $D_\eta^*$. Note that this distribution need *not* be centered at the risk minimizer of $c(w)$. What the result does, though, is establish a guarantee that each computer carrying out Algorithm 1 will converge rapidly to the same distribution over $w$, which will allow us to obtain good bounds if we can bound the 'bias' and 'variance' of $D_\eta^*$.

## 2.3   Guarantees for the Stationary Distribution

At this point, we know there exists a stationary distribution, and our algorithms are converging to that distribution exponentially fast. However, unlike in traditional gradient descent, the stationary distribution is not necessarily just the optimal point. In particular, the harder parts of understanding this algorithm involve understanding the properties of the stationary distribution. First, we show that the mean of the stationary distribution has low error. Therefore, if we ran for a really long time and averaged over many samples, the error would be low.

**Theorem 9** $c(\mathbf{E}_{w \in D_\eta^*}[w]) - \min_{w \in \mathbf{R}^n} c(w) \leq 2\eta G^2$.

Proven in Appendix G using techniques from regret minimization. Secondly, we show that the squared distance from the optimal point, and therefore the variance, is low.

**Theorem 10** *The average squared distance of $D_\eta^*$ from the optimal point is bounded by:*

$$\mathbf{E}_{w \in D_\eta^*}[(w - w^*)^2] \leq \frac{4\eta G^2}{(2 - \eta\lambda)\lambda}.$$

*In other words, the squared distance is bounded by $O(\eta G^2/\lambda)$.*

Proven in Appendix I using techniques from reinforcement learning. In what follows, if $x \in M$, $Y \in P(M, d)$, we define $W_z(x, Y)$ to be the $W_z$ distance between $Y$ and a distribution with a probability of 1 at $x$. Throughout the appendix, we develop tools to show that the distribution over the output vector of the algorithm is "near" $\mu_{D_\eta^*}$, the mean of the stationary distribution. In particular, if $D_\eta^{T,k}$ is the distribution over the final vector of ParallelSGD after $T$ iterations on each of $k$ machines with a learning rate $\eta$, then $W_2(\mu_{D_\eta^*}, D_\eta^{T,k}) = \sqrt{\mathbf{E}_{x \in D_\eta^{T,k}}[(x - \mu_{D_\eta^*})^2]}$ becomes small. Then, we need to connect the error of the mean of the stationary distribution to a distribution that is near to this mean.

**Theorem 11** *Given a cost function $c$ such that $\|c\|_L$ and $\|\nabla c\|_L$ are bounded, a distribution $D$ such that $\sigma_D$ and is bounded, then, for any $v$:*

$$\mathbf{E}_{w \in D}[c(w)] - \min_w c(w)$$

$$\leq (W_2(v, D))\sqrt{2 \|\nabla c\|_L (c(v) - \min_w c(w))} + \frac{\|\nabla c\|_L}{2}(W_2(v, D))^2 + (c(v) - \min_w c(w)). \quad (8)$$

This is proven in Appendix K. The proof is related to the Kantorovich-Rubinstein theorem, and bounds on the Lipschitz of $c$ near $v$ based on $c(v) - \min_w c(w)$. At this point, we are ready to get the *main theorem:*

**Theorem 12** *If $\eta \leq \eta^*$ and $T = \frac{\ln k - (\ln \eta + \ln \lambda)}{2\eta\lambda}$:*

$$\mathbf{E}_{w \in D_\eta^{T,k}}[c(w)] - \min_w c(w) \leq \frac{8\eta G^2}{\sqrt{k\lambda}}\sqrt{\|\nabla c\|_L} + \frac{8\eta G^2 \|\nabla c\|_L}{k\lambda} + (2\eta G^2). \quad (9)$$

This is proven in Appendix K.

## 2.4  Discussion of the Bound

The guarantee obtained in (9) appears rather unusual insofar as it does not have an explicit dependency on the sample size. This is to be expected since we obtained a bound in terms of risk minimization of the given corpus rather than a learning bound. Instead the runtime required depends only on the accuracy of the solution itself.

In comparison to [2], we look at the number of iterations to reach $\rho$ for SGD in Table 2. Ignoring the effect of the dimensions (such as $\nu$ and $d$), setting these parameters to 1, and assuming that the conditioning number $\kappa = \frac{1}{\lambda}$, and $\rho = \eta$. In terms of our bound, we assume $G = 1$ and $\|\nabla c\|_L = 1$. In order to make our error order $\eta$, we must set $k = \frac{1}{\lambda}$. So, the Bottou paper claims a bound of $\frac{\nu\kappa^2}{\rho}$ iterations, which we interpret as $\frac{1}{\eta\lambda^2}$. Modulo logarithmic factors, we require $\frac{1}{\lambda}$ machines to run $\frac{1}{\eta\lambda}$ time, which is the same order of computation, but a dramatic speedup of a factor of $\frac{1}{\lambda}$ in wall clock time.

Another important aspect of the algorithm is that it can be arbitrarily precise. By halving $\eta$ and roughly doubling $T$, you can halve the error. Also, the bound captures how much paralllelization can help. If $k > \frac{\|\nabla c\|_L}{\lambda}$, then the last term $\eta G^2$ will start to dominate.

## 3  Experiments

**Data:** We performed experiments on a proprietary dataset drawn from a major email system with labels $y \in \pm 1$ and binary, sparse features. The dataset contains $3,189,235$ time-stamped instances out of which the last $68,1015$ instances are used to form the test set, leaving $2,508,220$ training points. We used hashing to compress the features into a $2^{18}$ dimensional space. In total, the dataset contained $785,751,531$ features after hashing, which means that each instance has about 313 features on average. Thus, the average sparsity of each data point is 0.0012. All instance have been normalized to unit length for the experiments.

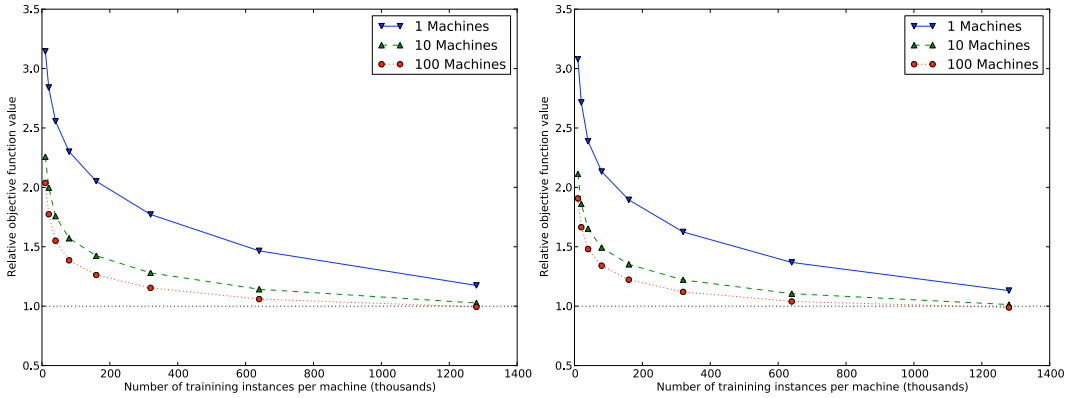

Figure 1: Relative training error with $\lambda = 1e^{-3}$: Huber loss (left) and squared error (right)

**Approach:** In order to evaluate the parallelization ability of the proposed algorithm, we followed the following procedure: For each configuration (see below), we trained up to 100 models, each on an independent, random permutation of the full training data. During training, the model is stored on disk after $k = 10,000 * 2^i$ updates. We then averaged the models obtained for each $i$ and evaluated the resulting model. That way, we obtained the performance for the algorithm after each machine has seen $k$ samples. This approach is geared towards the estimation of the parallelization ability of our optimization algorithm and its application to machine learning equally. This is in contrast to the evaluation approach taken in [7] which focussed solely on the machine learning aspect without studying the performance of the optimization approach.

**Evaluation measures:** We report both the normalized root mean squared error (RMSE) on the test set and the normalized value of the objective function during training. We normalize the RMSE such that $1.0$ is the RMSE obtained by training a model in one single, sequential pass over the data. The objective function values are normalized in much the same way such that the objective function value of a single, full sequential pass over the data reaches the value $1.0$.

**Configurations:** We studied both the Huber and the squared error loss. While the latter does not satisfy all the assumptions of our proofs (its gradient is unbounded), it is included due to its popularity. We choose to evaluate using two different regularization constants, $\lambda = 1e^{-3}$ and $\lambda = 1e^{-6}$ in order to estimate the performance characteristics both on smooth, "easy" problems ($1e^{-3}$) and on high-variance, "hard" problems ($1e^{-6}$). In all experiments, we fixed the learning rate to $\eta = 1e^{-3}$.

### 3.1 Results and Discussion

**Optimization:** Figure 1 shows the relative objective function values for training using $1, 10$ and $100$ machines with $\lambda = 1e^{-3}$. In terms of *wall clock time*, the models obtained on $100$ machines clearly outperform the ones obtained on $10$ machines, which in turn outperform the model trained on a single machine. There is no significant difference in behavior between the squared error and the Huber loss in these experiments, despite the fact that the squared error is effectively unbounded. Thus, the parallelization works in the sense that many machines obtain a better objective function value after each machine has seen $k$ instances. Additionally, the results also show that data-local parallelized training is feasible and beneficial with the proposed algorithm in practice. Note that the parallel training needs slightly more *machine time* to obtain the same objective function value, which is to be expected. Also unsurprising, yet noteworthy, is the trade-off between the number of machines and the quality of the solution: The solution obtained by $10$ machines is much more of an improvement over using one machine than using $100$ machines is over $10$.

**Predictive Performance:** Figure 2 shows the relative test RMSE for $1, 10$ and $100$ machines with $\lambda = 1e^{-3}$. As expected, the results are very similar to the objective function comparison: The parallel training decreases *wall clock time* at the price of slightly higher *machine time*. Again, the gain in performance between 1 and 10 machines is much higher than the one between 10 and 100.

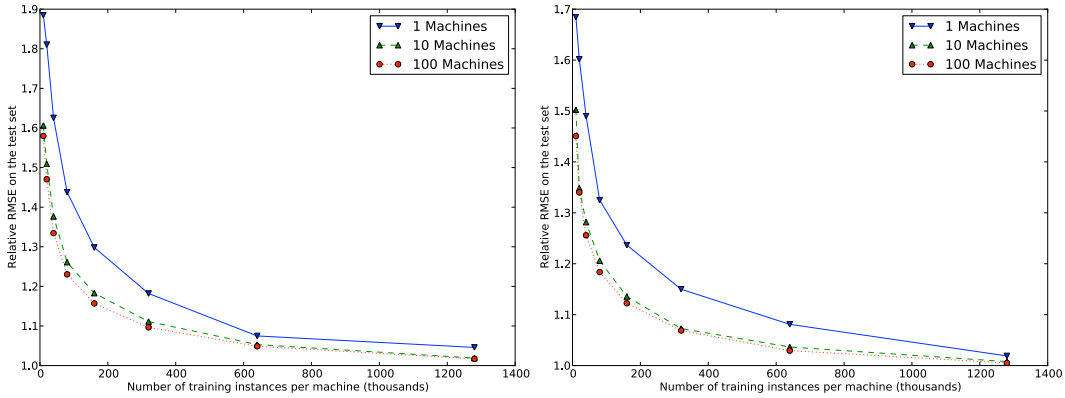

Figure 2: Relative Test-RMSE with $\lambda = 1e^{-3}$: Huber loss (left) and squared error (right)

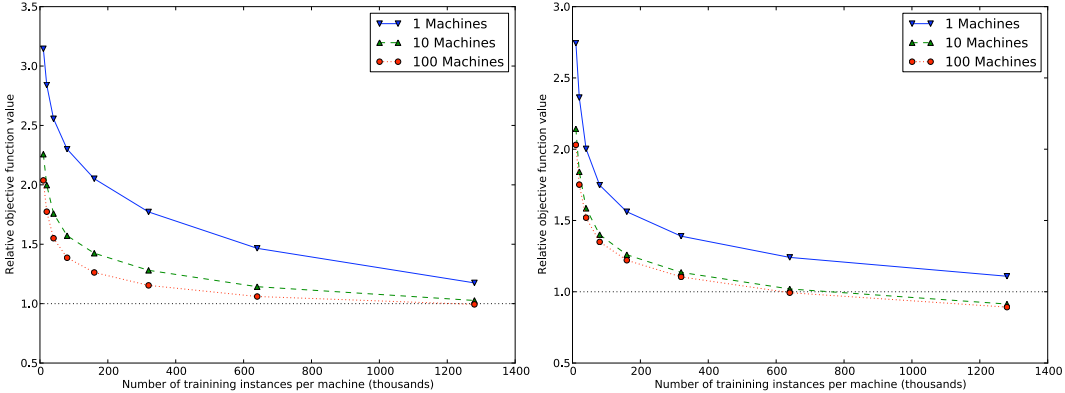

Figure 3: Relative train-error using Huber loss: $\lambda = 1e^{-3}$ (left), $\lambda = 1e^{-6}$ (right)

**Performance using different $\lambda$:** The last experiment is conducted to study the effect of the regularization constant $\lambda$ on the parallelization ability: Figure 3 shows the objective function plot using the Huber loss and $\lambda = 1e^{-3}$ and $\lambda = 1e^{-6}$. The lower regularization constant leads to more variance in the problem which in turn should increase the benefit of the averaging algorithm. The plots exhibit exactly this characteristic: For $\lambda = 1e^{-6}$, the loss for 10 and 100 machines not only drops faster, but the final solution for both beats the solution found by a single pass, adding further empirical evidence for the behaviour predicted by our theory.

## 4 Conclusion

In this paper, we propose a novel *data-parallel* stochastic gradient descent algorithm that enjoys a number of key properties that make it highly suitable for parallel, large-scale machine learning: It imposes very little I/O overhead: Training data is accessed locally and only the model is communicated at the very end. This also means that the algorithm is indifferent to I/O latency. These aspects make the algorithm an ideal candidate for a MapReduce implementation. Thereby, it inherits the latter's superb data locality and fault tolerance properties. Our analysis of the algorithm's performance is based on a novel technique that uses contraction theory to quantify finite-sample convergence rate of stochastic gradient descent. We show worst-case bounds that are comparable to stochastic gradient descent in terms of wall clock time, and vastly faster in terms of overall time. Lastly, our experiments on a large-scale real world dataset show that the parallelization reduces the wall-clock time needed to obtain a set solution quality. Unsurprisingly, we also see diminishing marginal utility of adding more machines. Finally, solving problems with more variance (smaller regularization constant) benefits more from the parallelization.

# References

[1] Shun-ichi Amari. A theory of adaptive pattern classifiers. *IEEE Transactions on Electronic Computers*, 16:299–307, 1967.

[2] L. Bottou and O. Bosquet. The tradeoffs of large scale learning. In *Advances in Neural Information Processing Systems*, 2008.

[3] C.T. Chu, S.K. Kim, Y. A. Lin, Y. Y. Yu, G. Bradski, A. Ng, and K. Olukotun. Map-reduce for machine learning on multicore. In B. Schölkopf, J. Platt, and T. Hofmann, editors, *Advances in Neural Information Processing Systems 19*, 2007.

[4] John Duchi, Elad Hazan, and Yoram Singer. Adaptive subgradient methods for online learning and stochastic optimization. In *Conference on Computational Learning Theory*, 2010.

[5] J. Langford, A.J. Smola, and M. Zinkevich. Slow learners are fast. In *Neural Information Processing Systems*, 2009.

[6] J. Langford, A.J. Smola, and M. Zinkevich. Slow learners are fast. arXiv:0911.0491, 2009.

[7] G. Mann, R. McDonald, M. Mohri, N. Silberman, and D. Walker. Efficient large-scale distributed training of conditional maximum entropy models. In Y. Bengio, D. Schuurmans, J. Lafferty, C. K. I. Williams, and A. Culotta, editors, *Advances in Neural Information Processing Systems 22*, pages 1231–1239. 2009.

[8] N. Murata, S. Yoshizawa, and S. Amari. Network information criterion—determining the number of hidden units for artificial neural network models. *IEEE Transactions on Neural Networks*, 5:865–872, 1994.

[9] Choon Hui Teo, S. V. N. Vishwanthan, Alex J. Smola, and Quoc V. Le. Bundle methods for regularized risk minimization. *J. Mach. Learn. Res.*, 11:311–365, January 2010.

[10] U. von Luxburg and O. Bousquet. Distance-based classification with lipschitz functions. *Journal of Machine Learning Research*, 5:669–695, 2004.

[11] M. Zinkevich. Online convex programming and generalised infinitesimal gradient ascent. In *Proc. Intl. Conf. Machine Learning*, pages 928–936, 2003.

